# Beware of Road Markings: A New Adversarial Patch Attack to Monocular Depth Estimation

**Hangcheng Liu[1], Zhenhu Wu[2], Hao Wang[3], Xingshuo Han[1]\*, Shangwei Guo[3], Tao Xiang[3], and Tianwei Zhang[1]**

[1]College of Computing and Data Science, Nanyang Technological University, Singapore
[2]School of Computer Science, Beijing University of Posts and Telecommunications, China
[3]College of Computer Science, Chongqing University, China
{hangcheng.liu, tianwei.zhang}@ntu.edu.sg
xingshuo001@e.ntu.edu.sg, wuzhenhu@bupt.edu.cn
{hwang, swguo, txiang}@cqu.edu.cn

## Abstract

Monocular Depth Estimation (MDE) enables the prediction of scene depths from a single RGB image, having been widely integrated into production-grade autonomous driving systems, e.g., Tesla Autopilot. Current adversarial attacks to MDE models focus on attaching an optimized adversarial patch to a designated obstacle. Although effective, this approach presents two inherent limitations: its reliance on specific obstacles and its limited malicious impact. In contrast, we propose a pioneering attack to MDE models that *decouples obstacles from patches physically and deploys optimized patches on roads*, thereby extending the attack scope to arbitrary traffic participants. This approach is inspired by our groundbreaking discovery: *various MDE models with different architectures, trained for autonomous driving, heavily rely on road regions* when predicting depths for different obstacles. Based on this discovery, we design the Adversarial Road Marking (`AdvRM`) attack, which camouflages patches as ordinary road markings and deploys them on roads, thereby posing a continuous threat within the environment. Experimental results from both dataset simulations and real-world scenarios demonstrate that `AdvRM` is effective, stealthy, and robust against various MDE models, achieving about 1.507 of Mean Relative Shift Ratio (MRSR) over 8 MDE models. The code is available at this Github Repo.

## 1 Introduction

Monocular Depth Estimation (MDE) [6, 7, 8, 9] is a technology that extracts depth information from monocular RGB images, enabling the projection of pixels from a 2D image into a 3D space. Due to its commendable efficiency and performance, MDE has been successfully employed in autonomous driving [10], e.g., Tesla's production-grade Autopilot system [11, 12, 13].

Recent studies [5, 14, 15] have demonstrated that MDE models are vulnerable to adversarial attacks, leading to erroneous depth predictions. Unlike global imperceptible adversarial perturbations [14, 15], which are effective only in the digital domain, adversarial patches [1, 2, 3, 4, 5, 16, 17] exhibit robust performance in the physical world, thus garnering significant attention. To our knowledge, existing patch attacks focus on single obstacle scenarios. They deploy an obstacle-dependent patch on a specific obstacle to attack the victim vehicle behind this obstacle, as illustrated in Fig. 1. These obstacle-dependent patches have two inherent limitations: 1) **limited impact**: the patch is only

---

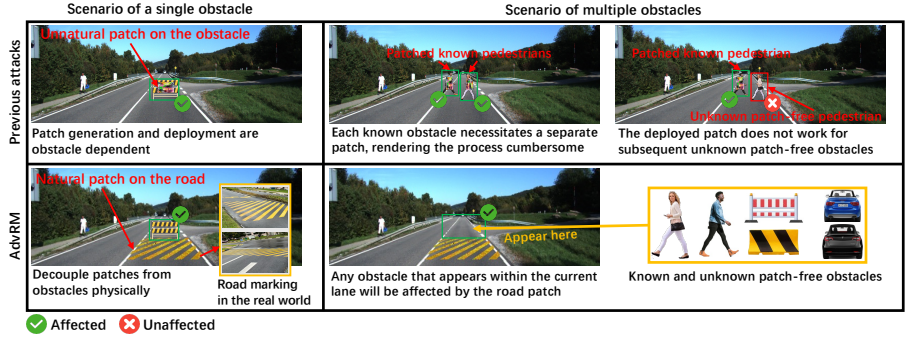

Figure 1: The difference between previous attacks (top) on MDE models and `AdvRM` (bottom). Previous attacks [1, 2, 3, 4, 5] are limited to single obstacle scenarios due to their obstacle-dependent patches. In contrast, the most significant advantage of `AdvRM` is **obstacle-agnostic**. This allows `AdvRM` to induce an MDE model to predict a false depth for any obstacle that appears in front of it, thereby being more suitable for complex multi-obstacle scenarios. Moreover, `AdvRM` **strategically places unobtrusive patches on the road**, exploiting the road dependency phenomenon we discovered in various mainstream MDE models, thereby making `AdvRM` both interpretable and applicable across different MDE models.

effective for the designated obstacle and cannot influence other unknown obstacles, making it challenging to adapt to complex traffic scenarios with multiple obstacles; and 2) **unstableness**: the effectiveness of these patches depends on the presence of the specific obstacles in the scene, rendering them unreliable for providing sustained threats, particularly those affixed on dynamic obstacles such as pedestrians. Table 1 reports more shortcomings of existing methods across various aspects.

Completely different from previous attacks against MDE, we make the **FIRST** attempt to decouple patches from obstacles physically, producing obstacle-independent adversarial patches and deploying them on roads (Fig. 1). By decoupling patches from obstacles, our road patches are no longer confined to altering the depth of specific known obstacles. Instead, the deployed road patch can mislead passing vehicles into changing the predicted depth of any obstacle that appears in front of it, even if the obstacle is unknown (Table 3), making it more suitable for complex traffic scenarios. Additionally, road patches typically maintain stability within a scene, thereby posing a persistent threat. Deploying patches on roads is inspired by our critical discovery: **MDE models trained for autonomous driving exhibit a strong dependency on roads when predicting depths for various obstacles** (Section 4). This suggests that road areas can serve as general patch areas for most MDE models. Moreover, we employ style transfer techniques [18] to disguise our road patches as visually innocuous road markings (Fig. 1), reducing suspicion while further increasing the patch's lifespan. Therefore, we call our attack Adversarial Road Marking (`AdvRM`).

Our main contributions can be summarized as follows:

- We conduct a comprehensive saliency analysis on various mainstream MDE models with different architectures and find that these models' predictions are road-dependent. We also provide a reasonable analysis of this phenomenon.

- We propose `AdvRM` which produces new road patches according to our observation and disguises them as ordinary road markings, misleading the passing victim vehicle's depth predictions for any possible obstacle.

- We conduct large-scale evaluations of the vulnerabilities of `AdvRM` across 3 CNN-based MDE models and 5 ViT-based MDE models in experiments.

## 2 Background and Related Works

### 2.1 Monocular Depth Estimation (MDE)

Existing MDE models can be divided into two categories according to their backbone architectures: CNN-based and ViT-based. CNNs were the preferred backbone for previous MDE studies [6, 7, 19,

Table 1: Comparisons between different patch attacks to MDE models. *Obstacle independent*: patches are independent of specific obstacles; *Semantic*: patches' appearance fit the context; *Stability*: patches pose a long-term threat; *Interpretability*: explanation for the selection of patch location

| Attack | Patch Location | Obstacle Independence | Multiple Obstacle | Affected Area | Semantic | Stability | Interpretability |
|---|---|---|---|---|---|---|---|
| [2] |  | ✗ | ✗ |  | ✗ | ✗ | ✗ |
| [3] |  | ✗ | ✗ |  | ✗ | ✗ | ✗ |
| [4] |  | ✗ | ✗ |  | ✔ | ✗ | ✗ |
| [1] |  | ✗ | ✗ |  | ✔ | ✗ | ✗ |
| [5] |  | ✗ | ✗ |  | ✔ | ✗ | ✗ |
| AdvRM (ours) |  | ✔ | ✔ | 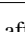 | ✔ | ✔ | ✔ |

 Obstacle    Partial areas of the 2D projection of obstacles are affected in depth maps.

 Road    All areas of the 2D projection of obstacles are affected in depth maps.

20, 21, 22]. However, Ranftl et al. [23] pointed out that the downsampling operations in CNNs limit dense predictions due to the resolution and granularity loss during the forward process. Therefore, many recent studies [8, 9, 24, 25, 26] start to adopt ViTs as the backbone, as they maintain feature resolution and enable global receptive fields through self-attention mechanisms. We notice that prior research on attacking MDE [1, 2, 3, 4, 4, 14, 15, 27] only focused on CNN-based MDE models, while largely neglecting ViT-based models. We fill this gap by demonstrating that ViT-based MDE models are also vulnerable to adversarial attacks.

## 2.2 Related Works and Comparisons

An attacker can employ carefully crafted adversarial perturbations (global perturbations or local patches) to alter the MDE models' predictions for targeted obstacles [1, 2, 3, 4, 5, 14, 15], compromising the depth perception of autonomous vehicles. Considering that adversarial patches [1, 2, 3, 4, 5] show better practicality in the physical world, we focus on patch attacks in this study. In Table 1, we show the major differences or advantages over the latest representative patch attacks targeting MDE. Evidently, our AdvRM exhibits several advantages. ❶ **Obstacle independence.** As AdvRM decouples patches from obstacles, the generation and deployment of patches are independent of specific obstacles, significantly increasing the adaptability of AdvRM in complex traffic scenarios, rendering it capable of changing the depth of unknown obstacles. In contrast, existing attacks focus on producing obstacle-dependent patches, which are less efficient or less effective in multi-obstacle scenarios, as shown in Fig. 1. ❷ **Stability.** In a traffic scene, obstacles are typically transient, with pedestrians and vehicles frequently entering and exiting, and even fixed roadblocks are generally temporary and eventually removed. Therefore, the life cycle of patches applied to specific obstacles is short-lived. On the contrary, our road patches, due to their natural appearance, are more likely to persist in traffic scenes for a longer period, posing a steady threat to passing vehicles and pedestrians. ❸ **Interpretability.** Previous attacks selected patch placement locations, such as the central region of the obstacle's 2D projection, without explaining the rationale. Cheng et al. [1] demonstrated that placing patches in different areas within the obstacle affects the attack's performance, yet failed to elucidate the optimal region choice from an interpretability perspective. In contrast, we conduct saliency analysis to identify the sensitivity of various mainstream MDE models to different regions of the environment (Section 4). Our findings reveal a consistent optimal patch region, i.e., the road, thereby providing good interpretability for our attack.

# 3 Preliminaries

## 3.1 Threat Model

We follow a similar adversarial scenario shown in [1, 2, 3], wherein an autonomous vehicle travels at a stable speed on lanes and uses an MDE model to perceive the depths of surrounding objects. To attack such an autonomous vehicle, an attacker carefully crafts an adversarial patch that looks like an ordinary road marking and places it in the path of a vehicle. This malicious patch forces the vehicle to estimate incorrect distances to obstacles in its front. The obstacles can be any traffic participants (e.g., cars, pedestrians) appearing there or static obstacles (e.g., roadblocks) placed in advance by the attacker. Incorrect depth information increases the likelihood of collisions.

We consider two specific goals for the attacker: (1) increasing the estimated depth of an obstacle, which may lead to delayed braking responses and potentially cause collisions with the obstacle. (2) Decreasing the estimated depth of an obstacle, which can result in phantom braking by the vehicle. Considering that the first goal has more serious consequences than the second, we mainly focus on increasing the estimated depth in the subsequent investigation. Nevertheless, our attack method can easily achieve a depth decrease as well.

Consistent with prior studies [1, 2, 3, 4, 15], we assume that the attacker possesses comprehensive knowledge regarding the target MDE model. It is practical because the attacker can rent a vehicle of the same model as the victim vehicle and engage in reverse engineering of the MDE model [12, 28].

## 3.2 Problem Formulation

Let $x \in \mathbb{R}^{3 \times h \times w}$ be a benign frame captured by a monocular camera. It can be represented as $x = b \otimes (1 - M_o) + o \otimes M_o$, where $o$ denotes obstacle, $b$ is environment (other areas besides $o$ within the frame), $\otimes$ is element-wise multiplication, and $M_o \in \mathbb{R}^{h \times w}$ is a binary mask, in which the obstacle area is filled with 1 and others are filled with 0. $f$ is an MDE model that outputs a depth map $f(x)$, representing relative depths rather than absolute depths. $\delta$ represents an adversarial patch capable of converting a benign environment $b$ into an adversarial environment $\hat{b}$ after being inserted into $b$. Let $\mathcal{A}_o$ and $\mathcal{A}_\delta$ represent two algorithms that insert given elements ($o$ or $\delta$) into the environment ($b$ or $\hat{b}$) according to the insertion parameters ($\theta_o$ or $\theta_\delta$). We have: $x = \mathcal{A}_o(b, o, \theta_o)$, $\hat{x} = \mathcal{A}_o(\hat{b}, o, \theta_o)$, and $\hat{b} = \mathcal{A}_\delta(b, \delta, \theta_\delta)$. For any $o$ and its $\theta_o$, we want to seek for an optimal $\delta$ and its $\theta_\delta$, satisfying

$$\text{mean}(f(\hat{x}) \otimes M_o) > \text{mean}(f(x) \otimes M_o), \tag{1}$$

where $\text{mean}(\cdot)$ indicates average function. We denote $f(x) \otimes M$ as $f_M(x)$ for simplification in subsequent descriptions, where $M$ represents arbitrary binary mask. In our study, $\theta_\delta$ is determined based on the characteristics of MDE models (Section 4.2). Therefore, given $b$ and $\theta_\delta$, we find a satisfactory $\delta$ by solving the following optimization:

$$\min_\delta \mathbb{E}_{o \sim p(o), \theta_o \sim p(\theta_o)} \left[ L_a(f_{M_o}(\hat{x}), f_{M_o}(x)) + \lambda \times L_{st}(\delta) \right], \tag{2}$$

where $p(o)$, $p(\theta_o)$, $L_a$, and $L_{st}$ denote the distributions of $o$ and $\theta_o$, adversarial loss, and stealthiness loss, respectively. $\lambda \geq 0$, balancing the attack effectiveness and stealthiness.

# 4 Saliency-driven Analysis for Patch Regions

## 4.1 Saliency Maps for MDE

Previous attacks propose placing patches at the center of obstacles [2, 3] or determining the patch position within the obstacle through optimization [1]. However, *it remains an unsolved question where the optimal patch regions in environments is.* Past works [29, 30] show that adding perturbations to regions that exert salient influence on the model predictions can increase the chance of successful attacks. We refer to these regions as *salient regions*, which can serve as candidates for deploying patches. Inspired by this, we employ saliency methods [31] to mark the salient regions of various MDE models and make a crucial observation. Saliency methods have emerged as popular tools to highlight features in an input that are closely related to the output, with gradient interpretation [32] serving as the foundation for many of these methods [33, 34, 35, 36]. Therefore, we employ gradient interpretation to define saliency maps $S$ in our study:

$$S = \frac{\sum_{i \in \{R, G, B\}} \left| \frac{\partial \, \text{mean}(f_M(x))}{\partial \, x[i]} \right|}{3}, \tag{3}$$

where $M$ here represents a binary mask, in which the designated local area is filled with 1 and others are filled with 0. $x[i]$ is one channel of RGB. A greater value in $S$ indicates that the predictive depth of the specified region is more responsive to the corresponding pixel in $x$.

## 4.2 Experiment Results

We use 8 state-of-the-art models as the target MDE models, including 3 CNN-based models (De-hin [20], Mono2 [6], Mande [7]) and 5 ViT-based models (Midas [8], Ada [37], GLPN [25],

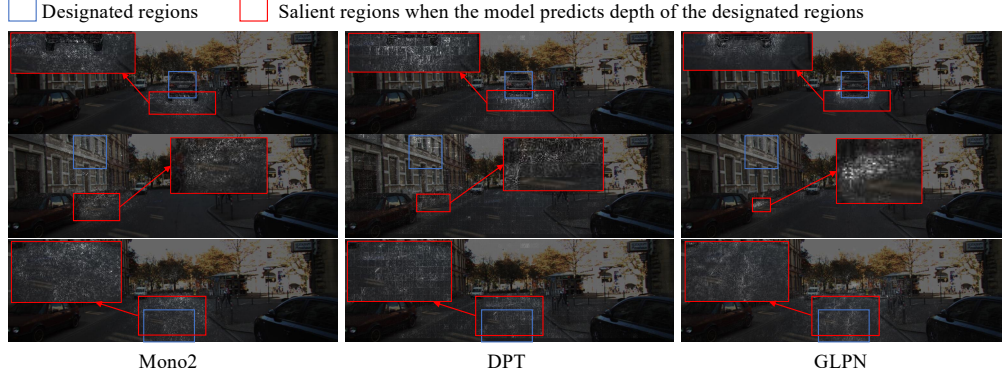

Figure 2: Saliency maps of different MDE models. The white points in the red box indicate that the depth prediction regarding the blue box strongly responds to the corresponding pixels.

DeAny [9], DPT [26]). All these models are trained on the KITTI dataset [38] or a hybrid dataset (consisting of various datasets). We randomly select 100 environment images from KITTI and gather obstacle images of cars, pedestrians, and roadblocks from Google Images.

We make an important finding from all generated $S$: **all these MDE models are road-dependent when predicting depths for various objects**. Fig. 2 illustrates this phenomenon, wherein we normalize the saliency maps within $[0, 1]$ and combine them with their input images by $x + \alpha \times S$ ($\alpha = 2.5$), thereby salient regions are filled with white points. We observe that *white points consistently appear within the road areas (red boxes) close to the designated regions (blue boxes)* despite the spatial separation between the road and the designated region (the second row in Fig. 2). This result indicates the significance of the road regions close to the designated area. Consequently, we decide to *deploy patches on the roads between the victim vehicles and obstacles strategically*.

### 4.3 Analysis

Before explaining the road-dependency, we first emphasize *the significance of perspective in depth estimation*. As widely acknowledged, perspective is a fundamental technique in artistic creation to imbue 2D images with a sense of depth, wherein distant objects appear smaller while nearer ones appear larger. Based on this principle, experienced individuals, e.g., snipers, can infer the distance of objects based on their observed scales and actual scales. This is also commonly presumed to be one of the underlying reasons why MDE models can estimate depths from monocular images [39, 40].

Most images from KITTI or similar datasets depict similar driving scenes wherein a lane occupies a central position, extending towards a central vanishing point, flanked by buildings, trees, or other objects along its periphery. Therefore, roads serve as natural vanishing points in these similar scenes akin to the auxiliary lines employed in artistic compositions, delineating the perspective relationships throughout the scene. All objects on the lane and those distributed along its peripheries conform to the perspective relationship delineated by the lane. So, *roads are the crucial cues to perspective*. Moreover, roads possess consistent widths across varying distances. Consequently, the width of the road can serve as a reliable reference for inferring the actual scale of different objects present on or near the road, enabling MDE models to estimate distances like an experienced sniper.

## 5 Methodology

### 5.1 Design Overview

Inspired by the road-dependent property we observed, we propose a new attack named Adversarial Road Marking (AdvRM), which strategically places unobtrusive adversarial patches on the road. Fig. 3 shows the overview of AdvRM. It mainly consists of three distinct steps: ❶ patch insertion, ❷ obstacle insertion, and ❸ patch optimization. In ❶, our proposed patch insertion algorithm $\mathcal{A}_\delta$ inserts $\delta$ into $b$ in a realistic manner based on four lane points (Fig. 4), which can be automatically annotated by lane detection methods [41]. In ❷, another insertion algorithm $\mathcal{A}_o$ randomly selects an obstacle image

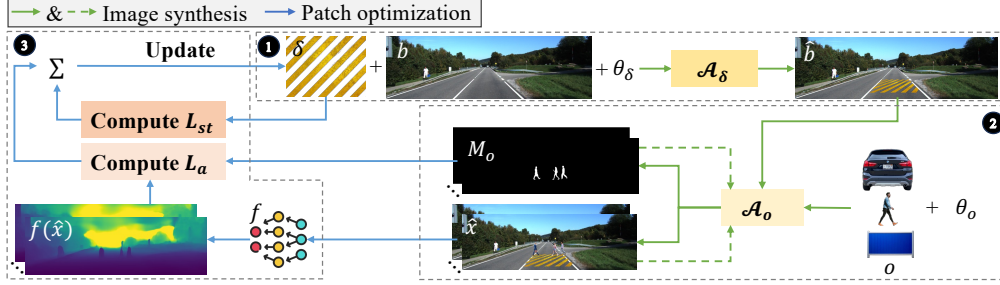

Figure 3: The overall pipeline of AdvRM consists of three main steps: ❶, ❷, and ❸, which correspond to patch insertion, obstacle insertion, and patch optimization, respectively. In step ❶, the patch insertion module $\mathcal{A}_\delta$ inserts the patch $\delta$ into the environment image based on parameters $\theta_\delta$. Similarly, in step ❷, the object insertion module $\mathcal{A}_o$ inserts the object $o$ into the image using parameters $\theta_o$. The green dashed lines in ❷ represent the process for inserting multiple obstacles into the same image. In step ❸, the weighted sum of $L_a$ and $L_{st}$ is computed, and $\delta$ is updated accordingly.

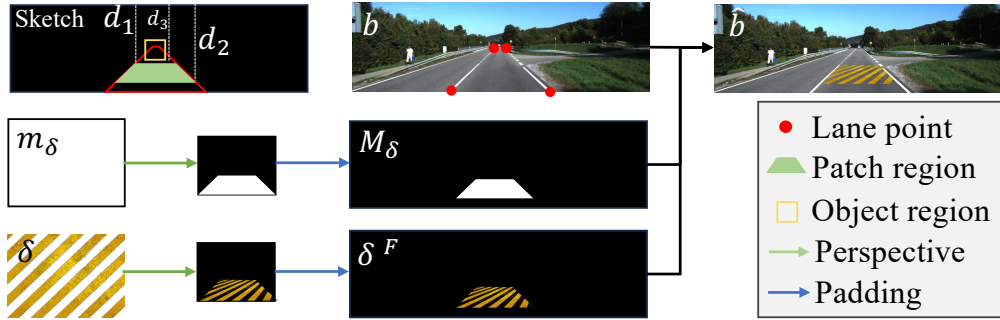

Figure 4: Insertion of patch. $d_1$, $d_2$, and $d_3$ are the distance from the upper and lower sides of the patch region and the lower edge of the object region to the top of the sketch. $M_\delta$ and $\delta^F$ are the same size as $b$.

from images of three kinds of common obstacles (cars, pedestrians, and roadblocks) and inserts it into $\hat{b}$, ensuring the patch applicability across diverse obstacles. Note that we can employ some predefined random image transformations (e.g., brightness, shifting, and rotation) in both ❶ and ❷ to process $\delta$ and $o$ to enhance the robustness of our attack. In ❸, we simultaneously minimize $L_a$ and $L_{st}$, balancing the effectiveness and stealthiness of our attack.

## 5.2 Image Synthesis

Image synthesis involves patch insertion and obstacle insertion, which requires reasonable perspective transformation and scaling for authentic synthesis. The necessary parameters for the two transformations can be calculated if both intrinsic and extrinsic camera parameters (e.g., the camera's focus length, height, and rotation) are known. Please refer to [1] for the details of the calculation.

We here introduce another effective way to complete authentic synthesis even if these intrinsic and extrinsic camera parameters are unclear. Our proposed insertion method relies on four key points of the central lane, as shown in Fig. 4. These key points can be annotated manually or automatically through lane detection [41]. We also manually check the annotation results and correct obvious errors. Based on these lane points, it is easy to sketch out the insertion (see the sketch in Fig. 4) according to the specified parameters ($d_1$, $d_2$, and $d_3$ in the sketch) using simple knowledge of analytic geometry. This sketch drives our two insertions ($\mathcal{A}_\delta$ and $\mathcal{A}_o$) to complete subsequent insertion. Specifically, $\mathcal{A}_\delta$ firsts employ a perspective function[2] to transform $\delta$ and its mask $m_\delta$ based on the vertices of the patch region in the sketch. The coordinates of the vertices can be easily calculated based on the lane points, $d_1$, and $d_2$. After that, $\mathcal{A}_\delta$ pads the transformed result with 0 to fit the dimensions of the input image. Finally, $\mathcal{A}_\delta$ synthesizes $\hat{b}$ by $\hat{b} = b \otimes (1 - M_\delta) + \delta^F \otimes M_\delta$. In the same way, $\mathcal{A}_o$ inserts $o$

into $\hat{b}$ according to lane points and $d_3$, producing $\hat{x}$ by $\hat{x} = \hat{b} \otimes (1 - M_o) + o^F \otimes M_o$, where $o^F$ and $M_o$, similar to $\delta^F$ and $M_\delta$ in Fig. 4, represent the padding results of the resized $o$ and its mask. For multi-obstacle scenarios, we mainly consider crowds (e.g., 3 pedestrians) crossing the road and apply a random horizontal offset to each inserted pedestrian. As for cars and roadblocks, we still focus on the case of a singular obstacle as they typically appear alone within a lane.

### 5.3 Robustness Enhancement

To enhance the robustness of `AdvRM` in the physical world, we can employ random transformations to process patches and obstacles during the insertion, which is known as Expectation of Transformation (EoT) [42]. Like most existing attacks [1, 3], we consider common brightness adjustments ($\pm 0.2$), contrast adjustments ($\pm 0.1$), and saturation adjustments ($\pm 0.1$). Additionally, to address the issue of road patches being obscured by fallen leaves or discarded plastic bags, we specifically introduce random pixel masking during the optimization to simulate this scenario.

### 5.4 Patch Optimization

As described in Eq. (2), we design loss function $L$ with the considerations of effectiveness and stealthiness, i.e., $L = L_a + \lambda \times L_{st}$. Adversarial loss $L_a$ ensures the capability of our deployed patch in manipulating depth maps, while stealthiness loss $L_{st}$ enhances stealthiness.

**Adversarial loss.** Most previous attacks [1, 2, 4, 5] focus solely on altering the depth of partial obstacle pixels within their adversarial loss, particularly those pixels overlapped by the patch. Such a design of adversarial loss results in limited effectiveness [3] and compromises attack stealthiness, as the depth variations within different parts of the same obstacle become conspicuously inconsistent, raising suspicion. Although Guesmi et al. [3] expands the affected area to the entire obstacle area using two loss terms corresponding to patch and non-patch regions, their method is not applicable here since all obstacle pixels are non-patch pixels in our attack scenario. Thus, we must revisit the design of adversarial loss.

To globally alter the depth of the obstacle and reduce the difference in depth changes across different obstacle regions, we categorize all obstacle pixels into two parts, easily affected pixels (EP) and less easily affected pixels (LEP), and pay more attention to LEP. EP and LEP are adaptively determined by the average change in the depth over all obstacle pixels. When the depth change exceeds the current average level, the corresponding pixels are regarded as EP; otherwise, the pixels are LEP. It is easy to understand that LEP should be given more attention to reduce the disparity with EP. We finally define our adversarial loss $L_a$ as

$$L_a = \beta \times L_{LEP} + L_{EP}, \text{ where}$$
$$L_\star = -\text{mean}(f_{M_o}(\hat{x}) \otimes M_\star), \star \in \{LEP, EP\},$$
$$M_\star = \begin{cases} \mathbb{1}(f_{M_o}(\hat{x}) < (f_{M_o}(x) \times \eta)), & \star = LEP \\ \mathbb{1}(f_{M_o}(\hat{x}) \geq (f_{M_o}(x) \times \eta)), & \star = EP \end{cases},$$
$$\eta = \max\left(1.14, \text{mean}\left(\frac{f_{M_o}(\hat{x})}{f_{M_o}(x)}\right)\right). \tag{4}$$

We have $\beta > 1$, and $\mathbb{1}$ is an index function and returns a mask filled with 0 and 1 according to the given condition. $M_{LEP}$ and $M_{EP}$ respectively identify LEP and EP. $\eta \geq 1.14$ where 1.14 is the minimum acceptable attack effectiveness of increased depth. Note that, we do not specify a target depth map in $L_a$ as in previous studies [1, 2, 3], opting instead to maximize the obstacle depth as much as possible due to our intention to explore the maximal efficacy of our new road patch.

**Stealthiness loss.** We consider attack stealthiness from two aspects: natural patch appearance and no noticeable change in the depth of the patched road area. To ensure the optimized patch looks like an ordinary road marking, we refer to [1] to build an appearance loss $L_{ap}$ using a deep photo style transfer proposed in [18]. $L_{ap}$ consists of four terms, i.e., $L_{ap} = L_s + L_c + L_t + L_r$, where $L_s$, $L_c$, $L_t$, and $L_r$ represent style loss, content loss, smoothness loss, and photorealism regularization loss, respectively. Please refer to Appendix for a detailed explanation of the four terms. On the other hand, a significant change in the depth of the patched road area will directly expose the existence of the malicious patch. Therefore, we employ another loss $L_{ma} = \text{mean}(|f_{M_\delta}(\hat{x}) - f_{M_\delta}(b)|)$ to maintain the depth of the patch area. Ultimately, our stealthiness loss $L_{st}$ is expressed as

$$L_{st} = L_{ap} + \sigma \times L_{ma}, (\sigma \geq 0). \tag{5}$$

Table 2: Performance of `AdvRM` when attacking different MDE models

| Metric | Obstacle | CNN | | | ViT | | | | |
|---|---|---|---|---|---|---|---|---|---|
| | | Dehin | Mono2 | Mande | Midas | Ada | GLPN | DeAny | DPT |
| $\xi_r$ | PE | 1.319 | 2.431 | 0.977 | 0.329 | 2.151 | 0.649 | 0.518 | 5.589 |
| | CA | 0.941 | 1.868 | 0.583 | 0.240 | 1.008 | 0.245 | 0.469 | 3.562 |
| | RO | 1.108 | 3.157 | 1.211 | 0.370 | 2.334 | 0.300 | 0.505 | 4.314 |
| | Average | 1.123 | 2.485 | 0.924 | 0.313 | 1.831 | 0.398 | 0.497 | 4.488 |
| $\xi_a$ | PE | 0.954 | 0.960 | 0.954 | 0.817 | 0.999 | 0.918 | 0.946 | 0.999 |
| | CA | 0.948 | 0.969 | 0.873 | 0.729 | 0.998 | 0.793 | 0.984 | 0.998 |
| | RO | 0.929 | 0.999 | 0.997 | 0.953 | 1.000 | 0.805 | 0.989 | 1.000 |
| | Average | 0.944 | 0.976 | 0.942 | 0.833 | 0.999 | 0.839 | 0.973 | 0.999 |

# 6 Experiments

## 6.1 Setup

**Dataset.** We randomly sample 100 images from KITTI [38] with different scenarios, which remains consistent with the previous study [1]. We manually annotate the lane points within these environment images to realize image synthesis. Additionally, we collect a total of 150 obstacle images of cars (CA), pedestrians (PE), and roadblocks (RO). We randomly split them into a training set (90 images) for optimizing patches and a test set (60 images) for evaluation. All synthesized input images are finally resized to a size of $320 \times 1024$.

**Implementation.** We align the patch's upper boundary with the obstacle's lower boundary, whose vertical positions are 230 ( the coordinate origin at the upper left corner of $\hat{x}$). In KITTI, this height corresponds to an approximate distance of 12 meters (m), the stopping distance requisite for a speed of 50 km/h [43], which is frequently employed in urban driving. We set the patch height to 70 pixels within $\hat{x}$, roughly equivalent to 4.5 m in the real world, while its width is set to the lane width. We set $\lambda = 100$ and $\sigma = 0.01$ when the target models are Dehin, Mono2, Mande, and DPT, and $\lambda = 50$ and $\sigma = 0.02$ otherwise. We set $\beta = 2$ for all MDE models. We choose BIM [44] to update $\delta$ with step size 0.01. The maximum number of iterations is 1000 when the target models are Midas and DPT; and 500 otherwise. All experiments are performed on a single GPU NVIDIA GeForce RTX 4090.

**Evaluation metrics.** All the selected models output relative depth maps instead of absolute depth maps. The scale of relative depth values varies heavily across different models. To address the scale problem and obtain comparable results, we design the Mean Relative Shift Ratio (MRSR) $\xi_r$ to measure the change in the obstacle depth before and after attacks. $\xi_r$ is defined as

$$\xi_r = \frac{\text{sum}(f_{M_o}(\hat{x}) - f_{M_o}(x))}{\text{sum}(f_{M_o}(x))}. \tag{6}$$

A positive $\xi_r$ means a farther predictive distance after attacks, while a negative $\xi_r$ indicates a closer depth. In our investigation, larger values of $\xi_r$ correspond to better attack performance. We also use Affect Region Ratio (ARR) [1], denoted as $\xi_a$, to measure the ratio of obstacle pixels whose $\xi_r$ exceeds a designated threshold. $\xi_a$ is defined as

$$\xi_a = \frac{\text{sum}\left(\mathbb{1}\left(f_{M_o}(\hat{x}) > (f_{M_o}(x) \times \eta_0)\right)\right)}{\text{sum}(M_o)}. \tag{7}$$

When $\xi_a$ approaches 1, we consider the attack's impact on obstacles to be global. Conversely, a low $\xi_a$ indicates a local depth alteration. We set $\eta_0 = 1.14$.

## 6.2 Dataset Simulation

**Effectiveness.**

Table 2 shows that `AdvRM` pose a significant security threat to these mainstream MDE models as it achieves high $\xi_r$ and $\xi_a$. Particularly, ***the average $\xi_r$ over all models is 1.507, indicating that an obstacle located at 12 m will be considered to be at 30 m, which is enough to delay braking and cause a serious collision.*** Fig. 5 confirms that `AdvRM` balances effectiveness and stealthiness well, where darker colors in the depth maps denote shorter distances while lighter colors correspond to

Table 3: Transferability of `AdvRM` across obstacles.

| Generation \ Test | Dehin | | | Mono2 | | | Mande | | | Midas | | |
|---|---|---|---|---|---|---|---|---|---|---|---|---|
| | PE | CA | RO | PE | CA | RO | PE | CA | RO | PE | CA | RO |
| PE | 1.178 | 0.330 | 0.419 | 3.174 | 1.312 | 2.626 | 1.426 | 0.362 | 0.660 | 0.320 | 0.180 | 0.263 |
| CA | 0.513 | 0.657 | 0.551 | 1.909 | 2.442 | 3.284 | 0.728 | 0.650 | 0.975 | 0.295 | 0.242 | 0.325 |
| RO | 0.930 | 0.732 | 1.030 | 2.694 | 2.204 | 4.559 | 1.184 | 0.680 | 1.617 | 0.296 | 0.237 | 0.337 |

| Generation \ Test | Ada | | | GLPN | | | DeAny | | | DPT | | |
|---|---|---|---|---|---|---|---|---|---|---|---|---|
| | PE | CA | RO | PE | CA | RO | PE | CA | RO | PE | CA | RO |
| PE | 3.366 | 1.000 | 2.072 | 1.188 | 0.057 | -0.011 | 0.471 | 0.419 | 0.461 | 6.954 | 1.460 | 3.219 |
| CA | 1.342 | 1.196 | 2.004 | 0.306 | 0.367 | 0.125 | 0.429 | 0.471 | 0.454 | 3.509 | 3.856 | 3.534 |
| RO | 1.903 | 1.072 | 2.646 | 0.320 | 0.226 | 0.457 | 0.191 | 0.293 | 0.452 | 4.216 | 2.593 | 5.221 |

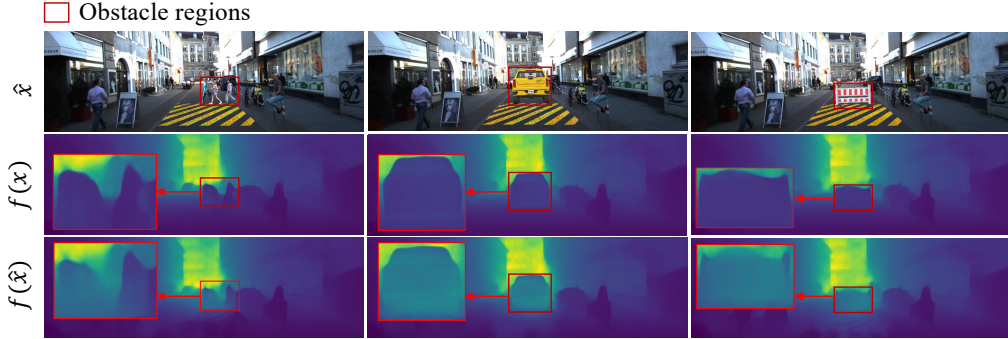

Figure 5: Patches on the road successfully change the predictive depth for different categories of obstacles. The color of the obstacle regions in $f(\hat{x})$ becomes lighter than that in $f(x)$, indicating larger predictive depths in $f(\hat{x})$.

longer distances. Meanwhile, `AdvRM` also performs well in situations of multiple obstacles because it makes the predicted depths of the three pedestrians farther simultaneously in Fig. 5.

**Robustness.** Fig. 6(a) reports the relative increments (%) in MRSR when employing EoT compared to its absence. The relative increment is defined as $\frac{\xi_r^w - \xi_r^{wo}}{\xi_r^{wo}} \times 100\%$, where $\xi_r^w$ and $\xi_r^{wo}$ denote the MRSR values of `AdvRM` when operating with and without EoT. All these increments confirm that EoT makes `AdvRM` more robust. Particularly, Fig. 6(b) shows that the optimized patches remain effective even when subjected to random masking, such as being partially covered by leaves or plastic bags (Fig. 6(c)).

**Obstacle transferability.** In this evaluation, we choose obstacles from one known category for patch generation, while the remaining two categories are unknown. Table 3 reports $\xi_r$ tested on known and unknown obstacles, confirming that `AdvRM` possesses a high transferability across obstacles. Therefore, ***our patches are capable of affecting whatever obstacles appear in front of it***.

## 6.3 Real-world Experiments

We also conduct physical experiments to demonstrate the robustness of `AdvRM` using printed roads, printed patches, and car models. Due to the limitations of testing environments, such a physical simulation is common in previous works [5].

**Sizes.** In the real world, a typical road width is 3.5 m, and the size of a Toyota Land Cruiser, a sport utility vehicle, is 4.95 m × 1.97 m × 1.905 m corresponding to length, width, and height. In the physical simulation, we scale these dimensions to approximately 1:50. Specifically, the printed lane width is scaled to 0.07 meters, with the patch dimensions measuring 0.06 meters in height and 0.07 meters in width. The dimensions of the scaled-down car model are 0.087 m × 0.035 m × 0.034 m.

**Results.** In this evaluation, we adopt LBFGS [45] to update the patch and use Mono2 [6] as the target model. As shown in Fig. 7, the difference between the farthest and closest distance is 0.06 m, corresponding to 3 m in the real world, and printed patch works for all frames, achieving an average $\xi_r$ of 1.088. Note that, the black stripes in the patch are only for ease of printing and pasting, containing no adversarial perturbations.

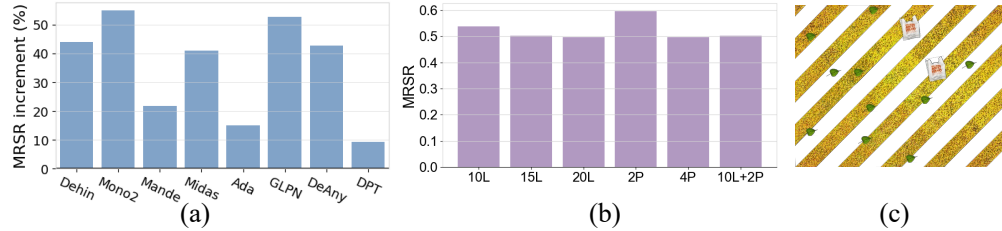

Figure 6: EoT enhances the robustness of `AdvRM`. (a) Increment on $\xi_r$ after adopting EoT. (b) Average $\xi_r$ of `AdvRM` when facing different numbers of covers, where 10L means 10 leaves, and 2P means 2 plastic bags. (c) Illustration of partial cover caused by leaves and plastic bags.

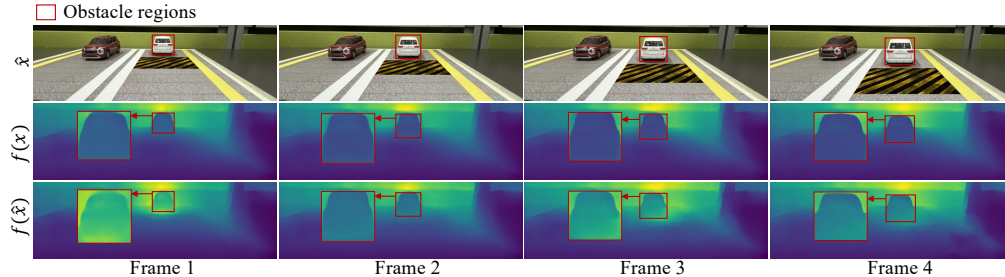

Figure 7: Testing results in the physical world. Our optimized patch simultaneously increases the predicted depth of the white car in all frames.

# 7 Limitations

Similar to existing works [5, 14, 15], `AdvRM` is also limited to white-box scenarios. In Table 4, we generate patches on surrogate models (e.g., DPT [26]) and use three state-of-the-art transferability enhancement methods, i.e., model ensemble [46] (ENS), gradient skipping [47] (GS), and gradient regularization [48] (GR), to improve the attack effectiveness on unknown victim models. However, it does not show obvious transferability across models. Indeed,

Table 4: Transferability across models measured by $\xi_r$. Values close to 0 indicate negligible model transferability.

| Obstacle | AdvRM | AdvRM-ENS | AdvRM-GS | AdvRM-GR |
|---|---|---|---|---|
| PE | 0.009 | 0.011 | 0.011 | 0.006 |
| CA | -0.023 | -0.014 | -0.018 | -0.023 |
| RO | -0.067 | -0.064 | -0.046 | -0.041 |

the transferability across MDE models remains an open problem, which will be a focal point for future research. Besides, `AdvRM` is currently only applicable to road scenarios. It is unclear whether the MDE models trained for other scenes (e.g., indoor) have similar road-dependent characteristics to support the decoupling of patches and obstacles.

# 8 Conclusion

We present a pioneering patch attack against MDE models that decouples adversarial patches from specified obstacles physically to broaden the applicability of attacks. This design is inspired by our crucial finding that current MDE models are road-dependent when predicting depths for obstacles. Based on this finding, we propose a new adversarial attack that places patches on the road between the vehicle and obstacles and disguises it as an ordinary road marking for high stealthiness. Experimental results from both dataset simulation and the physical world demonstrate `AdvRM` poses a serious threat to various MDE models as it significantly alters the depth predictions across different obstacle categories. Thus, we call on the community to pay more attention to the security issues of MDE models and actively propose measures to improve the robustness of MDE models.

# 9 Acknowledgements

We thank the anonymous reviewers for their valuable feedback. This work was supported in part by Nanyang Technological University (NTU)-DESAY SV Research Program under Grant 2018-0980,

National Research Foundation, Singapore and DSO National Laboratories under its AI Singapore Programme (AISG Award No: AISG2-GC-2023-008), National Natural Science Foundation of China under Grant 62072062, and Natural Science Foundation of Chongqing, China, under Grant cstc2022ycjh-bgzxm0031.

## Footnotes

[2]https://pytorch.org/vision/main/generated/torchvision.transforms.functional.perspective.html

[3]https://www.washingtonpost.com/lifestyle/2022/06/08/crosswalk-art-safety-bloomberg/

## References

[1] Zhiyuan Cheng, James Liang, Hongjun Choi, Guanhong Tao, Zhiwen Cao, Dongfang Liu, and Xiangyu Zhang. Physical attack on monocular depth estimation with optimal adversarial patches. In *ECCV*, 2022.

[2] Koichiro Yamanaka, Ryutaroh Matsumoto, Keita Takahashi, and Toshiaki Fujii. Adversarial patch attacks on monocular depth estimation networks. *IEEE Access*, 2020.

[3] Amira Guesmi, Muhammad Abdullah Hanif, Ihsen Alouani, and Muhammad Shafique. APARATE: Adaptive adversarial patch for cnn-based monocular depth estimation for autonomous navigation. *CoRR*, 2023.

[4] Amira Guesmi, Muhammad Abdullah Hanif, Bassem Ouni, and Muhammad Shafique. Saam: Stealthy adversarial attack on monocular depth estimation. *IEEE Access*, 2024.

[5] Junhao Zheng, Chenhao Lin, Jiahao Sun, Zhengyu Zhao, Qian Li, and Chao Shen. Physical 3d adversarial attacks against monocular depth estimation in autonomous driving. In *CVPR*, 2024.

[6] Clément Godard, Oisin Mac Aodha, Michael Firman, and Gabriel J Brostow. Digging into self-supervised monocular depth estimation. In *ICCV*, 2019.

[7] Jamie Watson, Oisin Mac Aodha, Victor Prisacariu, Gabriel Brostow, and Michael Firman. The temporal opportunist: Self-supervised multi-frame monocular depth. In *CVPR*, 2021.

[8] René Ranftl, Katrin Lasinger, David Hafner, Konrad Schindler, and Vladlen Koltun. Towards robust monocular depth estimation: Mixing datasets for zero-shot cross-dataset transfer. *TPAMI*, 2022.

[9] Lihe Yang, Bingyi Kang, Zilong Huang, Xiaogang Xu, Jiashi Feng, and Hengshuang Zhao. Depth anything: Unleashing the power of large-scale unlabeled data. *CoRR*, 2024.

[10] Markus Schön, Michael Buchholz, and Klaus Dietmayer. MGNet: Monocular geometric scene understanding for autonomous driving. In *ICCV*, 2021.

[11] Ai & Robotics, . `https://www.tesla.com/AI`.

[12] Hacker shows what tesla full self-driving's vision depth perception neural net can see, . `https://electrek.co/2021/07/07/hacker-tesla-full-self-drivings-vision-depth-perception-neural-net-can-see/`.

[13] Andrej karpathy. Ai for full-self driving at tesla. `https://www.youtube.com/watch?v=hx7BXih7zx8`.

[14] Ziqi Zhang, Xinge Zhu, Yingwei Li, Xiangqun Chen, and Yao Guo. Adversarial attacks on monocular depth estimation. *CoRR*, 2020.

[15] Alex Wong, Safa Cicek, and Stefano Soatto. Targeted adversarial perturbations for monocular depth prediction. *NeurIPS*, 2020.

[16] Xingshuo Han, Guowen Xu, Yuan Zhou, Xuehuan Yang, Jiwei Li, and Tianwei Zhang. Physical backdoor attacks to lane detection systems in autonomous driving. In *MM*, 2022.

[17] Yuan Xu, Xingshuo Han, Gelei Deng, Jiwei Li, Yang Liu, and Tianwei Zhang. Sok: Rethinking sensor spoofing attacks against robotic vehicles from a systematic view. In *EuroS&P*, 2023.

[18] Fujun Luan, Sylvain Paris, Eli Shechtman, and Kavita Bala. Deep photo style transfer. In *CVPR*, 2017.

[19] Clément Godard, Oisin Mac Aodha, and Gabriel J Brostow. Unsupervised monocular depth estimation with left-right consistency. In *CVPR*, 2017.

[20] Jamie Watson, Michael Firman, Gabriel J Brostow, and Daniyar Turmukhambetov. Self-supervised monocular depth hints. In *ICCV*, 2019.

[21] Ravi Garg, Vijay Kumar Bg, Gustavo Carneiro, and Ian Reid. Unsupervised cnn for single view depth estimation: Geometry to the rescue. In *ECCV*, 2016.

[22] Tinghui Zhou, Matthew Brown, Noah Snavely, and David G. Lowe. Unsupervised learning of depth and ego-motion from video. In *CVPR*, 2017.

[23] René Ranftl, Alexey Bochkovskiy, and Vladlen Koltun. Vision transformers for dense prediction. In *ICCV*, 2021.

[24] Shariq Farooq Bhat, Reiner Birkl, Diana Wofk, Peter Wonka, and Matthias Müller. Zoedepth: Zero-shot transfer by combining relative and metric depth. *CoRR*, 2023.

[25] Doyeon Kim, Woonghyun Ka, Pyungwhan Ahn, Donggyu Joo, Sehwan Chun, and Junmo Kim. Global-local path networks for monocular depth estimation with vertical cutdepth. *CoRR*, 2022.

[26] Maxime Oquab, Timothée Darcet, Théo Moutakanni, Huy Vo, Marc Szafraniec, Vasil Khalidov, Pierre Fernandez, Daniel Haziza, Francisco Massa, Alaaeldin El-Nouby, et al. Dinov2: Learning robust visual features without supervision. *CoRR*, 2023.

[27] Ibrahim Sobh, Ahmed Hamed, Varun Ravi Kumar, and Senthil Yogamani. Adversarial attacks on multi-task visual perception for autonomous driving. *CoRR*, 2021.

[28] Experimental security research of tesla autopilot, . https://keenlab.tencent.com/en/whitepapers/Experimental_Security_Research_of_Tesla_Autopilot.pdf.

[29] Tao Xiang, Hangcheng Liu, Shangwei Guo, Yan Gan, Wenjian He, and Xiaofeng Liao. Towards query-efficient black-box attacks: A universal dual transferability-based framework. *TIST*, 2023.

[30] Tianhang Zheng, Changyou Chen, Junsong Yuan, Bo Li, and Kui Ren. Pointcloud saliency maps. In *ICCV*, 2019.

[31] Julius Adebayo, Justin Gilmer, Michael Muelly, Ian Goodfellow, Moritz Hardt, and Been Kim. Sanity checks for saliency maps. *NeurIPS*, 31, 2018.

[32] Karen Simonyan, Andrea Vedaldi, and Andrew Zisserman. Deep inside convolutional networks: Visualising image classification models and saliency maps. *CoRR*, 2013.

[33] Ramprasaath R Selvaraju, Michael Cogswell, Abhishek Das, Ramakrishna Vedantam, Devi Parikh, and Dhruv Batra. Grad-cam: Visual explanations from deep networks via gradient-based localization. In *CVPR*, 2017.

[34] Avanti Shrikumar, Peyton Greenside, Anna Shcherbina, and Anshul Kundaje. Not just a black box: Learning important features through propagating activation differences. *CoRR*, 2016.

[35] Mukund Sundararajan, Ankur Taly, and Qiqi Yan. Axiomatic attribution for deep networks. In *ICML*, 2017.

[36] Daniel Smilkov, Nikhil Thorat, Been Kim, Fernanda Viégas, and Martin Wattenberg. Smooth-grad: removing noise by adding noise. *CoRR*, 2017.

[37] Shariq Farooq Bhat, Ibraheem Alhashim, and Peter Wonka. Adabins: Depth estimation using adaptive bins. In *CVPR*, 2021.

[38] Andreas Geiger, Philip Lenz, and Raquel Urtasun. Are we ready for autonomous driving? the kitti vision benchmark suite. In *CVPR*, 2012.

[39] Lubor Ladicky, Jianbo Shi, and Marc Pollefeys. Pulling things out of perspective. In *CVPR*, 2014.

[40] Yiran Wu, Sihao Ying, and Lianmin Zheng. Size-to-depth: A new perspective for single image depth estimation. In *CoRR*, 2018.

[41] Hiroto Honda and Yusuke Uchida. Clrernet: improving confidence of lane detection with laneiou. In *WACV*, 2024.

[42] Anish Athalye, Logan Engstrom, Andrew Ilyas, and Kevin Kwok. Synthesizing robust adversarial examples. In *ICML*, 2018.

[43] Stop distance. `http://www.csgnetwork.com/stopdistcalc.html`.

[44] Alexey Kurakin, Ian J Goodfellow, and Samy Bengio. Adversarial examples in the physical world. In *Artificial Intelligence Safety and Security*. 2018.

[45] Richard H Byrd, Peihuang Lu, Jorge Nocedal, and Ciyou Zhu. A limited memory algorithm for bound constrained optimization. *SIAM Journal on Scientific Computing*, 1995.

[46] Yifeng Xiong, Jiadong Lin, Min Zhang, John E Hopcroft, and Kun He. Stochastic variance reduced ensemble adversarial attack for boosting the adversarial transferability. In *CVPR*, 2022.

[47] Zhipeng Wei, Jingjing Chen, Micah Goldblum, Zuxuan Wu, Tom Goldstein, and Yu-Gang Jiang. Towards transferable adversarial attacks on vision transformers. In *AAAI*, 2022.

[48] Jianping Zhang, Yizhan Huang, Weibin Wu, and Michael R Lyu. Transferable adversarial attacks on vision transformers with token gradient regularization. In *CVPR*, 2023.

[49] Mahmood Sharif, Sruti Bhagavatula, Lujo Bauer, and Michael K Reiter. Accessorize to a crime: Real and stealthy attacks on state-of-the-art face recognition. In *ACM CCS*, 2016.

[50] Anat Levin, Dani Lischinski, and Yair Weiss. A closed-form solution to natural image matting. *TPAMI*, 2007.

# Appendix

## A  Obstacle Images Collection

We collect obstacle images of cars, pedestrians, and roadblocks, ensuring diversity within each category. For instance, our car images encompass various vehicle types (sedans, SUVs, trucks, etc.) and a range of colors. Fig. 8 illustrates a subset of the collected obstacle images.

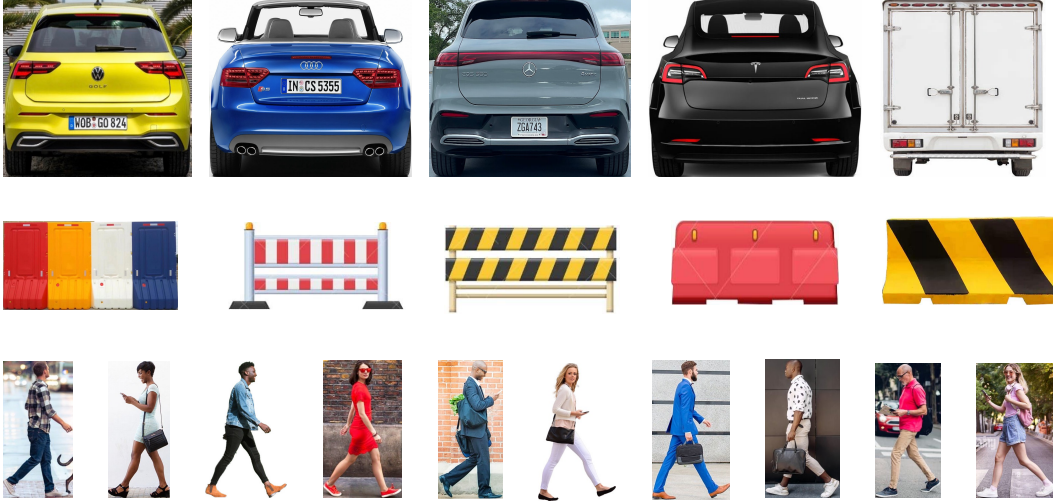

Figure 8: Collocted obstacle images.

## B  Stealthiness Loss

We detail the four items in the stealthiness loss $L_{ap}$. Let $H$ denote VGG19 trained on ImageNet, used for feature extraction. $I_s$ and $I_d$ represent a source image and a designated style image, respectively. In experiments, $I_s$ denotes the current patch which is initialized with $I_d$.

**Style loss.** The style loss is defined as:

$$L_s = \sum_{l \in K} \|G(H_l(I_s)) - G(H_l(I_d))\|_2^2, \tag{8}$$

where $H_l$ is the features at the $l$-th layer within $H$, $G$ represents the Gram matrix of the extracted features, and $K$ denotes the set of indexes of all selected layers. Specifically, $K = \{conv1\_1, conv2\_1, conv3\_1, conv4\_1, conv5\_1\}$ when calculating $L_s$. The smaller the value of $L_s$, the closer the style between the two images is.

**Content loss.** Similarly, the content loss is also defined based on the extracted features by $H$:

$$L_c = \sum_{l \in K} \|H_l(I_s) - H_l(I_d)\|_2^2. \tag{9}$$

Different from $L_s$, $L_c$ is calculated based on the Euclidean distance between the feature maps of $I_s$ and $I_d$. $K = \{conv4\_2\}$ when calculating $L_c$

**Smoothness loss.** This item encourages a locally smooth image, which improves stealthiness while also increasing patch robustness [49]. The smoothness loss is defined as:

$$L_t = \sum_{i,j} \left((I_s[i, j+1] - I_s[i,j])^2 + (I_s[i+1, j] - I_s[i,j])^2\right)^{\frac{1}{2}}, \tag{10}$$

where $I_s[i,j]$ denotes a pixel corresponding to the coordinate $(i,j)$.

**Photorealism regularization loss.** This loss is proposed in [18] for imposing certain constraints on color transfer, thereby preventing color distortions. It is defined as follows:

$$L_r = \sum_{c \in \{R,G,B\}} V_c(I_s)^\top \mathcal{M}(I_s) V_c(I_s), \tag{11}$$

where $c$ denotes one channel of RGB, $V_c$ reshapes its input into a shape of $N \times 1$ ($N$ represents the number of pixels in $I_s$), $\mathcal{M}(I_s) \in \mathbb{R}^{N \times N}$ represents a standard linear system that can minimize a least-square penalty function described in [50].

## C  Experiments

### C.1  Comparison With The State-Of-The-Art Attack

We compare `AdvRM` with [1] in single (one car) and multiple obstacles (three pedestrians) scenarios. Table 5 show the comparison results, in which `AdvRM` has higher MRSR and ARR than [1] in both scenarios. The reasons are that (a) [1] mainly affects the depth of the patch region while `AdvRM` affects all obstacle pixels, and (b) the patch in [1] only works for the known obstacle due to its obstacle-dependency while `AdvRM` is obstacle-agnostic.

Table 5: Comparison between `AdvRM` and [1] in single and multiple obstacles scenarios.

| Method | Metric | Single | Multiple |
|--------|--------|--------|----------|
| [1]    | $\xi_r$ | 1.019 | 0.136 |
|        | $\xi_a$ | 0.887 | 0.168 |
| AdvRM  | $\xi_r$ | 1.868 | 2.417 |
|        | $\xi_a$ | 0.969 | 0.958 |

### C.2  Evaluation Of Other Patching Regions

We further conduct ablation studies to show that MDE models rely significantly more on roads than other nearby areas. Specifically, we compare the attack performance of patches pasting onto different regions: (a) our patch on the road area as shown in Fig. 1; (b) a rectangular patch to the right of the obstacle, adjacent to but not occupying the center lane, like a roadside billboard; (c) a rectangular patch above the obstacle. The size of the right patch and top patch is $(p + 20) \times 70$, where $p$ is the width of the center lane, ensuring their sizes are similar to our road patch. For fair comparisons, we randomly choose Mono2 as the target model and keep other hyper-parameters the same. The experimental results are given in Table 6. We observe that the top and right patches are less effective than our road patch in altering the predicted depth, further supporting our insights regarding the road-dependent nature of MDE models.

Table 6: MRSR values when we place patches in different areas

| Obstacle | Top | Right | Road |
|----------|-----|-------|------|
| PE | 0.272 | 0.214 | 2.431 |
| CA | 0.411 | 0.292 | 1.868 |
| RO | 0.513 | 0.433 | 3.157 |

### C.3  Ablation Study

We employ three representative models, namely Midas [8], GLPN [25], and DeAny [9], to study the effect of patch sizes, spatial distance between patches and obstacles, and patch style.

**Threshold $\eta_0$.** We set $\eta_0 \geq 1.14$ based on our experimental setup to ensure that the prediction errors are sufficient to cause collisions. Specifically, means the model's predicted depth is off by at least 2.7 m when the ground truth is 12 m. Delayed braking caused by this error is enough to cause a minor collision at a vehicle speed of 50 km/h. Table 7 shows that `AdvRM` indeed realizes a high ARR since we do not limit the attack cap.

Table 7: ARR values of `AdvRM` under various $\eta_0$.

| $\eta_0$ | 1.14 | 1.2 | 1.3 | 1.4 | 1.5 |
|---|---|---|---|---|---|
| ARR | 0.976 | 0.974 | 0.965 | 0.951 | 0.887 |

**Patch size.** We vary the pixel height of patches between 30 and 70 in Fig. 4 while keeping the width consistent. In Fig. 9(a), despite the decline in patch size leading to a reduction in attack performance, `AdvRM` still achieves approximately 0.25 of $\xi_r$ at a patch height of 40.

**Spatial distance.** We introduce an upward offset, from 0 to 20 pixels, upon obstacles when inserting them. In Fig. 9(b), as the distance between the patch and obstacle increases, both $\xi_r$ and $\xi_a$ decline when attacking Midas and DeAny; however, they consistently maintain a satisfactory level when attacking GLPN.

**Patch style.** Fig. 10 shows four styles of patches, where P1, P2, and P3 are common road markings, and P4 is graffiti used as zebra crossings. Graffiti-style zebra crossings have appeared in many cities[3], providing attackers with exploitable opportunities because the variability in graffiti styles further enhances the stealthiness of attacks. Fig 9(c) exhibits that richly textured patches can improve attack performance.

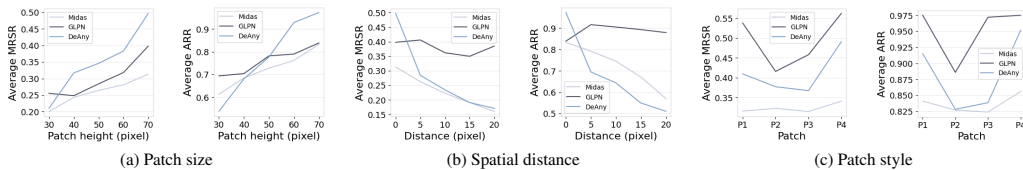

(a) Patch size          (b) Spatial distance          (c) Patch style

Figure 9: The impact of different factors on our attack.

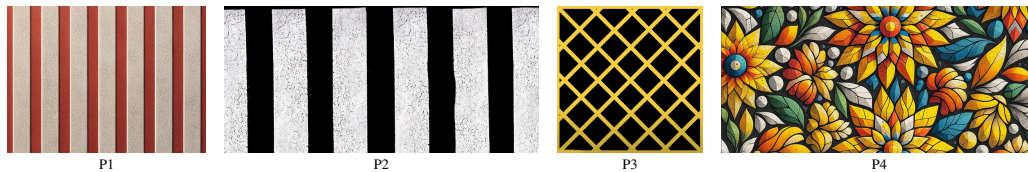

Figure 10: Other optional patch styles.

